# Information Processing to Create Eye Movements

**David A. Robinson**
Departments of Ophthalmology
and Biomedical Engineering
The Johns Hopkins University
School of Medicine
Baltimore, MD 21205

## ABSTRACT

Because eye muscles never cocontract and do not deal with external loads, one can write an equation that relates motoneuron firing rate to eye position and velocity - a very uncommon situation in the CNS. The semicircular canals transduce head velocity in a linear manner by using a high background discharge rate, imparting linearity to the premotor circuits that generate eye movements. This has allowed deducing some of the signal processing involved, including a neural network that integrates. These ideas are often summarized by block diagrams. Unfortunately, they are of little value in describing the behavior of single neurons - a finding supported by neural network models.

## 1 INTRODUCTION

The neural networks in our studies are quite simple. They differ from other applications in that they attempt to model real neural subdivisions of the oculomotor system which have been extensively studied with microelectrodes. Thus, we can ask the extent to which neural networks succeed in describing the behavior of hidden units that is already known. A major benefit of using neural networks in the oculomotor system is to illustrate clearly the shortcomings of block diagram models which tell one very little about what one may expect if one pokes a microelectrode inside one of its boxes. Conversely, single unit behavior is so loosely coupled to system behavior that, although the simplicity of the oculomotor system allows the relationships to be understood, one fears that, in a more complicated system, the behavior of single (hidden) units will give little information about what a system is trying to do, never mind how.

## 2  SIMPLIFICATIONS IN OCULOMOTOR CONTROL

Because it is impossible to cocontract our eye muscles and because their viscoelastic load never varies, it is possible to write an equation that uniquely relates the discharge rates of their motoneurons and the position of the load (eye position). This cannot be done in the case of, for example, limb muscles. Moreover, this system is well-approximated by a first-order, linear differential equation. Linearity comes about from the design of the semicircular canals, the origin of the vestibulo-ocular reflex (VOR). This reflex creates eye movements that compensate for head movements to stabilize the eyes in space for clear vision. The canals primarily transduce head velocity, neurally encoded into the discharge rates of its afferents. These rates modulate above and below a high background rate (typically 100 spikes/sec) that keeps them well away from cutoff and provides a wide linear range. The core of this reflex is only three neurons long and the canals impose their properties - linear modulation around a high background rate - onto all down-stream neurons including the motoneurons.

In addition to linearity, the functions of the various oculomotor subsystems are clear. There is no messy stretch reflex, the muscle fibers are straight and parallel, and there is only one "joint." All these features combine to help us understand the premotor organization of oculomotor signals in the caudal pons, a system that has enjoyed much block-diagram modelling and now, neural network modelling.

## 3  DISTRIBUTION OF OCULOMOTOR SIGNALS

The first application of neural networks to the oculomotor system was a study of Anastasio and Robinson (1989). The problem addressed concerned the convergence of diverse oculomotor signals in the caudal pons. There are three major oculomotor subsystems: the VOR; the saccadic system that causes the eyes to jump rapidly from one target to another; and the smooth pursuit system that allows the eyes to track a moving target. Each appears in the caudal pons as a velocity command. The canals, via the vestibular nuclei, provide an eye-velocity command, $E_v$, for compensatory vestibular eye movements. Burst neurons in the nearby pontine reticular formation provide a signal, $E_s$, for the desired eye velocity for a saccade. Purkinje cells in the cerebellum carry an eye-velocity signal, $E_p$, for pursuit eye movements. Thus, three eye-velocity commands converge in the region of the motoneurons.

When one records from cells in this region one finds a discharge rate R of:

$$R \simeq R_o + r_p \dot{E}_p + r_v \dot{E}_v + r_s \dot{E}_s \qquad (1)$$

where $R_o$ is the high background rate previously described and $r_p$, $r_v$ and $r_s$ are coefficients that can assume any values, in a seemingly random way, for any one neuron (e.g. Tomlinson and Robinson, 1984). Now a block-diagram model need show only the three velocity commands converging on the motoneurons and would not suggest the existence of neurons carrying complicated signals like that of Equ. (1). On the other hand, such behavior has a nice, messy, biological flavor. Somehow, it would seem odd if such signals did not exist. What is clearly happening is that the signals $E_p$, $E_v$ and $E_s$

are being distributed over the interneurons and then reassembled in the correct amount on the motoneurons. This is just a simple, specific example of distributed parallel processing in the nervous system.

A neural network model is merely an explicit statement of such a distribution. Initial randomization of the synaptic weights followed by error-driven learning creates hidden units that conform to Equ. (1). We concluded that a neural network model was entirely appropriate for this neural system. This exercise also brought home, although in a simple way, the obvious, but often overlooked, message that block-diagram models can be misleading about how their conceptual functions are realized by neurons.

We next examined distribution of the spatial properties of the interneurons of the VOR (Anastasio and Robinson, 1990). We used only the vertical VOR to keep things simple. The inputs were the primary afferents of the four vertical semicircular canals that sense head rotations in all combinations of pitch and roll. The output layer was the four motoneurons of the vertical recti and oblique muscles that move the eye vertically and in cyclotorsion. The model was trained to perform compensatory eye movements in all combinations of pitch and roll.

The sensitivity axis is that axis around which rotation of the head or eye produces maximum modulation in discharge rate. The sensitivity axis of a canal unit is perpendicular to the plane in which the canal lies. That of a motoneuron is that axis around which its muscle will rotate the eye. What were the sensitivity axes of the hidden units?

A block diagram of the spatial manipulations of the VOR consists of matrices. The geometry of the canals can be described by a 3 x 3 matrix that converts a head-velocity vector into its neurally encoded representation on canal nerves. The geometry of the muscles can be described as another matrix that converts the neurally-encoded motoneuron vector into a physical eye-rotation vector. The brain-stem matrix describes how the canal neurons must project to the motoneurons (Robinson, 1982). In this scheme, interneurons would have only fixed sensitivity axes laying somewhere between that of a canal unit and a motoneuron. In our model, however, sensitivity axes are distributed in the network; those of the hidden units point in a variety of directions. This has also been confirmed by microelectrode recordings (Fukushima et al., 1990). Thus, spatial aspects of transformations, just like temporal aspects, are distributed over the interneurons.

Again, block-diagrams, in this case in the form of a matrix, are misleading about what one will find with a microelectrode. Again, recording from single units tells one little about what a network is trying to do. There is much talk in motor physiology about coordinate systems and transformations from one to another. The question is asked "What coordinate system is this neuron working in?" In this example, individual hidden units do not behave as if they belonged to any coordinate system and this raises the problem of whether this is really a meaningful question.

## 4 THE NEURAL INTEGRATOR

Muscles are largely position actuators; against a constant load, position is proportional

to innervation. The motoneurons of the extraocular muscles also need a signal proportional to desired eye position as well as velocity. Since eye-movement commands enter the caudal pons as eye-velocity commands, the necessary eye-position command is obtained by integrating the velocity signals (see Robinson, 1989, for a review). The location of the neural network has been discovered in the caudal pons and it is intriguing to speculate how it might work. Hardwired networks, based on positive feedback, have been proposed utilizing lateral inhibition (Cannon et al., 1983) and more recently a learning neural network (dynamic) has been proposed for the VOR (Arnold and Robinson, 1991). The hidden units are freely connected, the input is from two canal units in push-pull, the output is two motoneurons also in push-pull, which operate on the plant transfer function, $1/(sT_e + 1)$, ($T_e$ is the plant time constant), to create an eye position which should be the time integral of the input head velocity. The error is retinal image slip (the difference between actual and ideal eye velocity). Its rms value over a trial interval is used to change synaptic weights in a steepest descent method until the error is negligible. To compensate the plant lag, the network must produce a combination output of eye velocity plus its integral, eye position, and these two signals, with various weights, are seen on all hidden units which, thus, look remarkably like the integrator neurons that we record from.

This exercise raises several issues. The block-diagram model of this network is a box marked 1/s in parallel with the direct velocity feedforward path given the gain $T_e$. The parallel combination is $(sT_e + 1)/s$. The zero cancels the pole of the plant leaving 1/s, so that eye position is the perfect integral of head velocity. While such a diagram is conceptually very useful in diagnosing disorders (Zee and Robinson, 1979), it contains no hint of how neurons might effect integration and so is useless in this regard. Moreover, Galiana and Outerbridge (1984) have pointed out, although in a more complex context, that a direct feedforward path of gain $T_e$ with a positive feedback path around it containing a model of the plant, produces exactly the same transfer function. Should we worry about which is correct - feedforward or feedback? Perhaps we should, but note that the neural network model of the integrator just described contains both feedback and feedforward pathways and relies on positive feedback. There is a suspicion that the latter network may subsume both block diagrams making questions about which is correct irrelevant. One thing is certain, at this level of organization, so close to the neuron level, block-diagrams, while having conceptual value, are not only useless but can be misleading if one is interested in describing real neural networks.

Finally, how does one test a model network such as that proposed for the neural integrator? It involves the microcircuitry with which small sets of circumscribed cells talk to each other and process signals. The technology is not yet available to allow us to answer this question. I know of no real, successful examples. This, I believe, is a true roadblock in neurophysiology. If we cannot solve it, we must forever be content to describe what cell groups do but not how they do it.

### Acknowledgements
This research is supported by Grant 5 R37 EY00598 from the National Eye Institute of the National Institutes of Health.

## References

T.J. Anastasio & D.A. Robinson. (1989) The distributed representation of vestibulo-ocular signals by brain-stem neurons. *Biol. Cybern.*, **61**:79-88.

T.J. Anastasio & D.A. Robinson. (1990) Distributed parallel processing in the vertical vestibulo-ocular reflex: Learning networks compared to tensor theory. *Biol. Cybern.*, **63**:161-167.

D.B. Arnold & D.A. Robinson. (1991) A learning network model of the neural integrator of the oculomotor system. *Biol. Cybern.*, **64**:447-454.

S.C. Cannon, D.A. Robinson & S. Shamma. (1983) A proposed neural network for the integrator of the oculomotor system. *Biol. Cybern.*, **49**:127-136.

K. Fukushima, S.I. Perlmutter, J.F. Baker & B.W. Peterson. (1990) Spatial properties of second-order vestibulo-ocular relay neurons in the alert cat. *Exp. Brain Res.*, **81**:462-478.

H.L. Galiana & J.S. Outerbridge. (1984) A bilateral model for central neural pathways in vestibuloocular reflex. *J. Neurophysiol.*, **51**:210-241.

D.A. Robinson. (1982) The use of matrices in analyzing the three-dimensional behavior of the vestibulo-ocular reflex. *Biol. Cybern.*, **46**:53-66.

D.A. Robinson. (1989) Integrating with neurons. *Ann. Rev. Neurosci.*, **12**:33-45.

R.D. Tomlinson & D.A. Robinson. (1984) Signals in vestibular nucleus mediating vertical eye movements in the monkey. *J. Neurophysiol.*, **51**:1121-1136.

D.S. Zee & D.A. Robinson. (1979) Clinical applications of oculomotor models. In H.S. Thompson (ed.), *Topics in Neuro-Ophthalmology*, 266-285. Baltimore, MD: Williams & Wilkins.
